# Finding Structure in Reinforcement Learning

**Sebastian Thrun**
University of Bonn
Department of Computer Science III
Römerstr. 164, D-53117 Bonn, Germany
E-mail: thrun@carbon.informatik.uni-bonn.de

**Anton Schwartz**
Dept. of Computer Science
Stanford University
Stanford, CA 94305
Email: schwartz@cs.stanford.edu

## Abstract

Reinforcement learning addresses the problem of learning to select actions in order to maximize one's performance in unknown environments. To scale reinforcement learning to complex real-world tasks, such as typically studied in AI, one must ultimately be able to discover the structure in the world, in order to abstract away the myriad of details and to operate in more tractable problem spaces.

This paper presents the SKILLS algorithm. SKILLS discovers skills, which are partially defined action policies that arise in the context of multiple, related tasks. Skills collapse whole action sequences into single operators. They are learned by minimizing the compactness of action policies, using a description length argument on their representation. Empirical results in simple grid navigation tasks illustrate the successful discovery of structure in reinforcement learning.

## 1  Introduction

Reinforcement learning comprises a family of incremental planning algorithms that construct reactive controllers through real-world experimentation. A key scaling problem of reinforcement learning, as is generally the case with unstructured planning algorithms, is that in large real-world domains there might be an enormous number of decisions to be made, and pay-off may be sparse and delayed. Hence, instead of learning all single fine-grain actions all at the same time, one could conceivably learn much faster if one abstracted away the myriad of micro-decisions, and focused instead on a small set of important decisions. But this immediately raises the problem of how to recognize the important, and how to distinguish it from the unimportant.

This paper presents the SKILLS algorithm. SKILLS finds partially defined action policies, called skills, that occur in more than one task. Skills, once found, constitute parts of solutions to multiple reinforcement learning problems. In order to find maximally useful skills, a description length argument is employed. Skills reduce the number of bytes required to describe action policies. This is because instead of having to describe a complete action policy for each task separately, as is the case in plain reinforcement learning, skills constrain multiple task to pick the same actions, and thus reduce the total number of actions required

for representing action policies. However, using skills comes at a price. In general, one cannot constrain actions to be the same in multiple tasks without ultimately suffering a loss in performance. Hence, in order to find maximally useful skills that infer a minimum loss in performance, the SKILLS algorithm minimizes a function of the form

$$E \;=\; PERFORMANCE\ LOSS \;+\; \eta \cdot DESCRIPTION\ LENGTH. \tag{1}$$

This equation summarizes the rationale of the SKILLS approach. The reminder of this paper gives more precise definitions and learning rules for the terms *"PERFORMANCE LOSS"* and *"DESCRIPTION LENGTH,"* using the vocabulary of reinforcement learning. In addition, experimental results empirically illustrate the successful discovery of skills in simple grid navigation domains.

## 2   Reinforcement Learning

Reinforcement learning addresses the problem of learning, through experimentation, to act so as to maximize one's pay-off in an unknown environment. Throughout this paper we will assume that the environment of the learner is a *partially controllable Markov chain* [1]. At any instant in time the learner can observe the state of the environment, denoted by $s \in S$, and apply an action, $a \in A$. Actions change the state of the environment, and also produce a scalar pay-off value, denoted by $r_{s,a} \in \Re$. Reinforcement learning seeks to identify an *action policy*, $\pi : S \longrightarrow A$, i.e., a mapping from states $s \in S$ to actions $a \in A$ that, if actions are selected accordingly, maximizes the *expected discounted sum of future pay-off*

$$R \;=\; E\left[\sum_{t=t_0}^{\infty} \gamma^{t-t_0}\ r_t\right]. \tag{2}$$

Here $\gamma$ (with $0 \leq \gamma \leq 1$) is a *discount factor* that favors pay-offs reaped sooner in time, and $r_t$ refers to the expected pay-off at time $t$. In general, pay-off might be delayed. Therefore, in order to learn an optimal $\pi$, one has to solve a *temporal credit assignment problem* [11].

To date, the single most widely used algorithm for learning from delayed pay-off is Q-Learning [14]. Q-Learning solves the problem of learning $\pi$ by learning a *value function*, denoted by $Q : S \times A \longrightarrow \Re$. $Q$ maps states $s \in S$ and actions $a \in A$ to scalar values. After learning, $Q(s, a)$ ranks actions according to their goodness: The larger the expected cumulative pay-off for picking action $a$ at state $s$, the larger the value $Q(s, a)$. Hence $Q$, once learned, allows to maximize $R$ by picking actions greedily with respect to $Q$:

$$\pi(s) \;=\; \underset{\hat{a} \in A}{\operatorname{argmax}}\ Q(s, \hat{a})$$

The value function $Q$ is learned on-line through experimentation. Initially, all values $Q(s, a)$ are set to zero. Suppose during learning the learner executes action $a$ at state $s$, which leads to a new state $s'$ and the immediate pay-off $r_{s,a}$. Q-Learning uses this state transition to update $Q(s, a)$:

$$Q(s, a) \;\longleftarrow\; (1 - \alpha) \cdot Q(s, a) \;+\; \alpha \cdot (r_{s,a} + \gamma \cdot V(s')) \tag{3}$$

$$\text{with}\quad V(s') \;=\; \max_{\hat{a}} Q(s', \hat{a})$$

The scalar $\alpha$ ($0 < \alpha \leq 1$) is the *learning rate*, which is typically set to a small value that is decayed over time. Notice that if $Q(s, a)$ is represented by a lookup-table, as will be the case throughout this paper, the Q-Learning rule (3) has been shown[1] to converge to a value function $Q^{\mathrm{opt}}(s, a)$ which measures the future discounted pay-off one can expect to receive upon applying action $a$ in state $s$, and acting optimally thereafter [5, 14]. The greedy policy $\pi(s) = \operatorname{argmax}_{\hat{a}} Q^{\mathrm{opt}}(s, \hat{a})$ maximizes $R$.

## 3 Skills

Suppose the learner faces a whole collection of related tasks, denoted by $B$, with identical states $S$ and actions $A$. Suppose each task $b \in B$ is characterized by its individual pay-off function, denoted by $r_b(s, a)$. Different tasks may also face different state transition probabilities. Consequently, each task requires a task-specific value function, denoted by $Q_b(s, a)$, which induces a task-specific action policy, denoted by $\pi_b$. Obviously, plain Q-Learning, as described in the previous section, can be employed to learn these individual action policies. Such an approach, however, cannot discover the structure which might inherently exist in the tasks.

In order to identify commonalities between different tasks, the SKILLS algorithm allows a learner to acquire *skills*. A skill, denoted by $k$, represents an action policy, very much like $\pi_b$. There are two crucial differences, however. Firstly, skills are only locally defined, on a subset $S_k$ of all states $S$. $S_k$ is called the *domain of skill k*. Secondly, skills are not specific to individual tasks. Instead, they apply to entire sets of tasks, in which they replace the task-specific, local action policies.

Let $K$ denote the set of all skills. In general, some skills may be appropriate for some tasks, but not for others. Hence, we define a vector of *usage values* $u_{k,b}$ (with $0 \leq u_{k,b} \leq 1$ for all $k \in K$ and all $b \in B$). Policies in the SKILLS algorithm are stochastic, and usages $u_{k,b}$ determine how frequently skill $k$ is used in task $b$. At first glance, $u_{k,b}$ might be interpreted as a probability for using skill $k$ when performing task $b$, and one might always want to use skill $k$ in task $b$ if $u_{k,b} = 1$, and never use skill $k$ if $u_{k,b} = 0$.[2] However, skills might overlap, *i.e.*, there might be states $s$ which occurs in several skill domains, and the usages might add to a value greater than 1. Therefore, usages are normalized, and actions are drawn probabilistically according to the normalized distribution:

$$P_b(k|s) \quad = \quad \frac{u_{b,k}^2 \cdot m_k(s)}{\sum_{k' \in K} u_{b,k'} \cdot m_{k'}(s)} \qquad \text{(with } \tfrac{0}{0} = 0) \tag{4}$$

Here $P_b(k|s)$ denotes the *probability* for using skill $k$ at state $s$, if the learner faces task $b$. The indicator function $m_k(s)$ is the *membership function* for skill domains, which is 1 if $s \in S_k$ and 0 otherwise. The probabilistic action selection rule (4) makes it necessary to redefine the value $V_b(s)$ of a state $s$. If no skill dictates the action to be taken, actions will be drawn according to the $Q_b$-optimal policy

$$\pi_b^*(s) \quad = \quad \underset{\hat{a} \in A}{\text{argmax}}\, Q_b(s, \hat{a}) \,,$$

as is the case in plain Q-Learning. The probability for this to happen is

$$P_b^*(s) \quad = \quad 1 - \sum_{k \in K} P_b(k|s) \,.$$

Hence, the *value* of a state is the weighted sum

$$V_b(s) \quad = \quad P_b^*(s) \cdot V_b^*(s) + \sum_{k \in K} P_b(k|s) \cdot Q_b(s, \pi_k(s)) \tag{5}$$

$$\text{with } V_b^*(s) \quad = \quad Q_b(s, \pi_b^*(s)) \quad = \quad \max_{\hat{a} \in A} Q_b(s, \hat{a})$$

Why should a learner use skills, and what are the consequences? Skills reduce the freedom to select actions, since multiple policies have to commit to identical actions. Obviously, such

a constraint will generally result in a *loss in performance*. This loss is obtained by comparing the actual value of each state $s$, $V_b(s)$, and the value if no skill is used, $V_b^*(s)$:

$$LOSS = \sum_{s \in S} \underbrace{\sum_{b \in B} V_b^*(s) - V_b(s)}_{= LOSS(s)} \tag{6}$$

If actions prescribed by the skills are close to optimal, *i.e.*, if $V_b^*(s) \approx V_b(s)(\forall s \in S)$, the loss will be small. If skill actions are poor, however, the loss can be large.

Counter-balancing this loss is the fact that skills give a more compact representation of the learner's policies. More specifically, assume (without loss of generality) actions can be represented by a single byte, and consider the total number of bytes it takes to represent the policies of all tasks $b \in B$. In the absence of skills, representing all individual policies requires $|B| \cdot |S|$ bytes, one byte for each state in $S$ and each task in $B$. If skills are used across multiple tasks, the description length is reduced by the amount of overlap between different tasks. More specifically, the total description length required for the specification of all policies is expressed by the following term:

$$DL = \sum_{s \in S} \sum_{b \in B} P_b^*(s) + \sum_{k \in K} |S_k| = \sum_{s \in S} \underbrace{\left( \sum_{b \in B} P_b^*(s) + \sum_{k \in K} m_k(s) \right)}_{= DL(s)} \tag{7}$$

If all probabilities are binary, *i.e.*, $P_b(k|s)$ and $P_b^*(s) \in \{0, 1\}$, $DL$ measures precisely the number of bytes needed to represent all skill actions, plus the number of bytes needed to represent task-specific policy actions where no skill is used. Eq. (7) generalizes this measure smoothly to stochastic policies. Notice that the number of skills $|K|$ is assumed to be constant and thus plays no part in the description length $DL$.

Obviously, minimizing $LOSS$ maximizes the pay-off, and minimizing $DL$ maximizes the compactness of the representation of the learner's policies. In the SKILLS approach, one seeks to minimize both (*cf.* Eq. (1))

$$E = LOSS + \eta DL = \sum_{s \in S} LOSS(s) + \eta DL(s). \tag{8}$$

$\eta > 0$ is a gain parameter that trades off both target functions. $E$-optimal policies make heavily use of large skills, yet result in a minimum loss in performance. Notice that the state space may be partitioned completely by skills, and solutions to the individual tasks can be uniquely described by the skills and its usages. If such a complete partitioning does not exist, however, tasks may instead rely to some extent on task-specific, local policies.

## 4   Derivation of the Learning Algorithm

Each skill $k$ is characterized by three types of adjustable variables: *skill actions* $\pi_k(s)$, the *skill domain* $S_k$, and *skill usages* $u_{b,k}$, one for each task $b \in B$. In this section we will give update rules that perform hill-climbing in $E$ for each of these variables. As in Q-Learning these rules apply only at the currently visited state (henceforth denoted by $s$). Both learning action policies (*cf.* Eq. (3)) and learning skills is fully interleaved.

**Actions.** Determining skill actions is straightforward, since what action is prescribed by a skill exclusively affects the performance loss, but does not play any part in the description length. Hence, the action policy $\pi_k(s)$ minimizes $LOSS(s)$ (*cf.* Eqs. (5) and (6)):

$$\pi_k(s) = \underset{\hat{a} \in A}{argmax} \sum_{b \in B} P_b(k|s) \cdot Q_b(s, \hat{a}) \tag{9}$$

**Domains.** Initially, each skill domain $S_k$ contains only a single state that is chosen at random. $S_k$ is changed incrementally by minimizing $E(s)$ for states $s$ which are visited during learning. More specifically, for each skill $k$, it is evaluated whether or not to include $s$ in $S_k$ by considering $E(s) = LOSS(s) + \eta DL(s)$.

$$s \in S_k, \quad \text{if and only if} \quad E(s)|_{s \in S_k} < E(s)|_{s \notin S_k} \qquad \text{(otherwise } s \notin S_k) \qquad (10)$$

If the domain of a skill $k$ vanishes completely, *i.e.*, if $S_k = \emptyset$, it is re-initialized by a randomly selected state. In addition all usage values $\{u_{b,k}|b \in B\}$ are initialized randomly. This mechanism ensures that skills, once overturned by other skills, will not get lost forever.

**Usages.** Unlike skill domains, which are discrete quantities, usages are real-valued numbers. Initially, they are chosen at random in $[0, 1]$. Usages are optimized by stochastic gradient descent in $E$. According to Eq. (8), the derivative of $E(s)$ is the sum of $\frac{\partial LOSS(s)}{\partial u_{b,k}}$ and $\frac{\partial DL(s)}{\partial u_{b,k}}$. The first term is governed by

$$\frac{\partial LOSS(s)}{\partial u_{b,k}} = -\frac{\partial V_b(s)}{\partial u_{b,k}} = -\frac{\partial P_b^*(s)}{\partial u_{b,k}} \cdot Q_b(\pi_b^*(s), s) - \sum_{j \in K} \frac{\partial P_b(j|s)}{\partial u_{b,k}} \cdot Q_b(s, \pi_j(s))$$

$$\text{with} \quad \frac{\partial P_b(j|s)}{\partial u_{b,k}} = m_j(s) \cdot \left( \frac{2\delta_{kj} u_{b,j}}{\sum_{k' \in K} m_{k'}(s) u_{b,k'}} - \frac{u_{b,j}^2 m_k(s)}{\left( \sum_{k' \in K} m_{k'}(s) u_{b,k'} \right)^2} \right) \qquad (11)$$

$$\text{and} \quad \frac{\partial P_b^*(s)}{\partial u_{b,j}} = -\sum_{j \in K} \frac{\partial P_b(j|s)}{\partial u_{b,j}}. \qquad (12)$$

Here $\delta_{kj}$ denotes the Kronecker delta function, which is 1 if $k = j$ and 0 otherwise. The second term is given by

$$\frac{\partial DL(s)}{\partial u_{b,k}} = \frac{\partial P_b^*(s)}{\partial u_{k,b}}, \qquad (13)$$

which can be further transformed using Eqs. (12) and (11). In order to minimize $E$, usages are incrementally refined in the opposite direction of the gradients:

$$u_{k,b} \longleftarrow u_{k,b} - \beta \cdot \left( \frac{\partial V(s)}{\partial u_{k,b}} + \eta \frac{\partial DL(s)}{\partial u_{k,b}} \right) \qquad (14)$$

Here $\beta > 0$ is a small learning rate. This completes the derivation of the SKILLS algorithm. After each action execution, Q-Learning is employed to update the Q-function. SKILLS also re-calculates, for any applicable skill, the skill policy according to Eq. (9), and adjusts skill domains and usage values based upon Eqs. (10) and (14).

## 5  Experimental Results

The SKILLS algorithm was applied to discover skills in a simple, discrete grid-navigation domain, depicted in Fig. 1. At each state, the agent can move to one of at most eight adjacent grid cells. With a 10% chance the agent is carried to a random neighboring state, regardless of the commanded action. Each corner defines a starting state for one out of four task, with the corresponding goal state being in the opposite corner. The pay-off (costs) for executing actions is $-1$, except for the goal state, which is an absorbing state with zero pay-off. In a first experiment, we supplied the agent with two skills $K = \{k_1, k_2\}$. All four tasks were trained in a time-shared manner, with time slices being 2,000 steps long. We used the following parameter settings: $\eta = 1.2$, $\gamma = 1$, $\alpha = 0.1$, and $\beta = 0.001$.

After 30 000 training steps for each task, the SKILLS algorithm has successfully discovered the two skills shown in Figure 1. One of these skills leads the agent to the right door, and

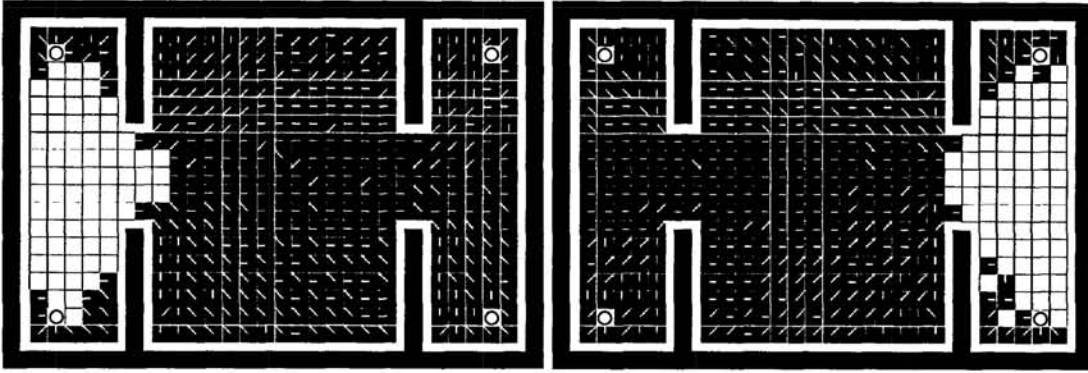

Figure 1: Simple 3-room environment. Start and goal states are marked by circles. The diagrams also shows two skills (black states), which lead to the doors connecting the rooms.

---

the second to the left. Each skill is employed by two tasks. By forcing two tasks to adopt a single policy in the region of the skill, they both have to sacrifice performance, but the loss in performance is considerably small. Beyond the door, however, optimal actions point into opposite directions. There, forcing both tasks to select actions according to the same policy would result in a significant performance loss, which would clearly outweigh the savings in description length. The solution shown in Fig. 1 is (approximately) the global minimum of $E$, given that only two skills are available. It is easy to be seen that these skills establish helpful building blocks for many navigation tasks.

When using more than two skills, $E$ can be minimized further. We repeated the experiment using six skills, which can partition the state space in a more efficient way. Two of the resulting skills were similar to the skills shown in Fig. 1, but they were defined only between the doors. The other four skills were policies for moving out of a corner, one for each corner. Each of the latter four skills can be used in three tasks (unlike two tasks for passing through the middle room), resulting in an improved description length when compared to the two-skill solution shown in Fig. 1.

We also applied skill learning to a more complex grid world, using 25 skills for a total of 20 tasks. The environment, along with one of the skills, is depicted in Fig. 2. Different tasks were defined by different starting positions, goal positions and door configurations which could be open or closed. The training time was typically an order of magnitude slower than in the previous task, and skills were less stable over time. However, Fig. 2 illustrates that modular skills could be discovered even in such complex a domain.

## 6  Discussion

This paper presents the SKILLS algorithm. SKILLS learns skills, which are partial policies that are defined on a subset of all states. Skills are used in as many tasks as possible, while affecting the performance in these tasks as little as possible. They are discovered by minimizing a combined measure, which takes a task performance and a description length argument into account.

While our empirical findings in simple grid world domains are encouraging, there are several open questions that warrant future research.

**Learning speed.** In our experiments we found that the time required for finding useful skills is up to an order of magnitude larger than the time it takes to find close-to-optimal policies.

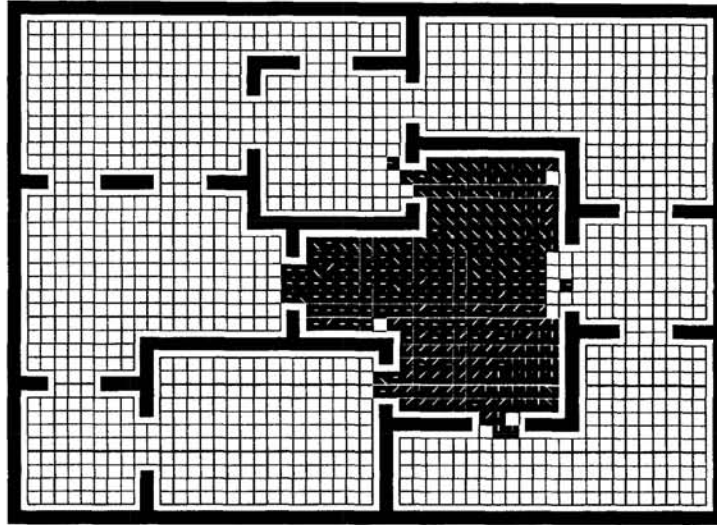

Figure 2: Skill found in a more complex grid navigation task.

Similar findings are reported in [9]. This is because discovering skills is much harder than learning control. Initially, nothing is know about the structure of the state space, and unless reasonably accurate $Q$-tables are available, SKILLS cannot discover meaningful skills. Faster methods for learning skills, which might precede the development of optimal value functions, are clearly desirable.

**Transfer.** We conjecture that skills can be helpful when one wants to learn new, related tasks. This is because if tasks are related, as is the case in many natural learning environments, skills allow to transfer knowledge from previously learned tasks to new tasks. In particular, if the learner faces tasks with increasing complexity, as proposed by Singh [10], learning skills could conceivable reduce the learning time in complex tasks, and hence scale reinforcement learning techniques to more complex tasks.

**Using function approximators.** In this paper, performance loss and description length has been defined based on table look-up representations of $Q$. Recently, various researchers have applied reinforcement learning in combination with generalizing function approximators, such as nearest neighbor methods or artificial neural networks (*e.g.*, [2, 4, 12, 13]). In order to apply the SKILLS algorithm together with generalizing function approximators, the notions of skill domains and description length have to be modified. For example, the membership function $m_k$, which defines the domain of a skill, could be represented by a function approximator which allows to derive gradients in the description length.

**Generalization in state space.** In the current form, SKILLS exclusively discovers skills that are used across multiple tasks. However, skills might be useful under multiple circumstances even in single tasks. For example, the (generalized) skill of climbing a staircase may be useful several times in one and the same task. SKILLS, in its current form, cannot represent such skills.

The key to learning such generalized skills is generalization. Currently, skills generalize exclusively over tasks, since they can be applied to entire sets of tasks. However, they cannot generalize over states. One could imagine an extension to the SKILLS algorithm, in which skills are free to pick what to generalize over. For example, they could chose to ignore certain state information (like the color of the staircase). It remains to be seen if effective learning mechanisms can be designed for learning such generalized skills.

**Abstractions and action hierarchies.** In recent years, several researchers have recognized the importance of structuring reinforcement learning in order to build abstractions and action

hierarchies. Different approaches differ in the origin of the abstraction, and the way it is incorporated into learning. For example, abstractions have been built upon previously learned, simpler tasks [9, 10], previously learned low-level behaviors [7], subgoals, which are either known in advance [15] or determined at random [6], or based on a pyramid of different levels of perceptual resolution, which produces a whole spectrum of problem solving capabilities [3]. For all these approaches, drastically improved problem solving capabilities have been reported, which are far beyond that of plain, unstructured reinforcement learning. This paper exclusively focuses on how to discover the structure inherent in a family of related tasks. Using skills to form abstractions and learning in the resulting abstract problem spaces is beyond the scope of this paper. The experimental findings indicate, however, that skills are powerful candidates for operators on a more abstract level, because they collapse whole action sequences into single entities.

## Footnotes

[1] under certain conditions concerning the exploration scheme, the environment and the learning rate

[2]This is exactly the action selection mechanism in the SKILLS algorithm if only *one* skill is applicable at any given state $s$.

# References

[1] A. G. Barto, S. J. Bradtke, and S. P. Singh. Learning to act using real-time dynamic programming. *Artificial Intelligence*, to appear.

[2] J. A. Boyan. Generalization in reinforcement learning: Safely approximating the value function. Same volume.

[3] P. Dayan and G. E. Hinton. Feudal reinforcement learning. In J. E. Moody, S. J. Hanson, and R. P. Lippmann, editors, *Advances in Neural Information Processing Systems 5*, 1993. Morgan Kaufmann.

[4] V. Gullapalli, J. A. Franklin, and Hamid B. Acquiring robot skills via reinforcement learning. *IEEE Control Systems*, 272(1708), 1994.

[5] T. Jaakkola, M. I. Jordan, and S. P. Singh. On the convergence of stochastic iterative dynamic programming algorithms. Technical Report 9307, Department of Brain and Cognitive Sciences, MIT, July 1993.

[6] L. P. Kaelbling. Hierarchical learning in stochastic domains: Preliminary results. In Paul E. Utgoff, editor, *Proceedings of the Tenth International Conference on Machine Learning*, 1993. Morgan Kaufmann.

[7] L.-J. Lin. *Self-supervised Learning by Reinforcement and Artificial Neural Networks*. PhD thesis, Carnegie Mellon University, School of Computer Science, 1992.

[8] M. Ring. Two methods for hierarchy learning in reinforcement environments. In *From Animals to Animates 2: Proceedings of the Second International Conference on Simulation of Adaptive Behavior*. MIT Press, 1993.

[9] S. P. Singh. Reinforcement learning with a hierarchy of abstract models. In *Proceeding of the Tenth National Conference on Artificial Intelligence AAAI-92*, 1992. AAAI Press/The MIT Press.

[10] S. P. Singh. Transfer of learning by composing solutions for elemental sequential tasks. *Machine Learning*, 8, 1992.

[11] R. S. Sutton. *Temporal Credit Assignment in Reinforcement Learning*. PhD thesis, Department of Computer and Information Science, University of Massachusetts, 1984.

[12] G. J. Tesauro. Practical issues in temporal difference learning. *Machine Learning*, 8, 1992.

[13] S. Thrun and A. Schwartz. Issues in using function approximation for reinforcement learning. In M. Mozer, Pa. Smolensky, D. Touretzky, J. Elman, and A. Weigend, editors, *Proceedings of the 1993 Connectionist Models Summer School*, 1993. Erlbaum Associates.

[14] C. J. C. H. Watkins. *Learning from Delayed Rewards*. PhD thesis, King's College, Cambridge, England, 1989.

[15] S. Whitehead, J. Karlsson, and J. Tenenberg. Learning multiple goal behavior via task decomposition and dynamic policy merging. In J. H. Connell and S. Mahadevan, editors, *Robot Learning*. Kluwer Academic Publisher, 1993.
